# Optimal Kernel Shapes for Local Linear Regression

**Dirk Ormoneit**        **Trevor Hastie**
Department of Statistics
Stanford University
Stanford, CA 94305-4065
*ormoneit@stat.stanford.edu*

## Abstract

Local linear regression performs very well in many low-dimensional forecasting problems. In high-dimensional spaces, its performance typically decays due to the well-known "curse-of-dimensionality". A possible way to approach this problem is by varying the "shape" of the weighting kernel. In this work we suggest a new, data-driven method to estimating the optimal kernel shape. Experiments using an artificially generated data set and data from the UC Irvine repository show the benefits of kernel shaping.

## 1 Introduction

Local linear regression has attracted considerable attention in both statistical and machine learning literature as a flexible tool for nonparametric regression analysis [Cle79, FG96, AMS97]. Like most statistical smoothing approaches, local modeling suffers from the so-called "curse-of-dimensionality", the well-known fact that the proportion of the training data that lie in a fixed-radius neighborhood of a point decreases to zero at an exponential rate with increasing dimension of the input space. Due to this problem, the bandwidth of a weighting kernel must be chosen very big so as to contain a reasonable sample fraction. As a result, the estimates produced are typically highly biased. One possible way to reduce the bias of local linear estimates is to vary the "shape" of the weighting kernel. In this work, we suggest a method for estimating the optimal kernel shape using the training data. For this purpose, we parameterize the kernel in terms of a suitable "shape matrix", $L$, and minimize the mean squared forecasting error with respect to $L$. For such an approach to be meaningful, the "size" of the weighting kernel must be constrained during the minimization to avoid overfitting. We propose a new, entropy-based measure of the kernel size as a constraint. By analogy to the nearest neighbor approach to bandwidth selection [FG96], the suggested measure is adaptive with regard to the local data density. In addition, it leads to an efficient gradient descent algorithm for the computation of the optimal kernel shape. Experiments using an artificially generated data set and data from the UC Irvine repository show that kernel shaping can improve the performance of local linear estimates substantially.

The remainder of this work is organized as follows. In Section 2 we briefly review

local linear models and introduce our notation. In Section 3 we formulate an objective function for kernel shaping, and in Section 4 we discuss entropic neighborhoods. Section 5 describes our experimental results and Section 6 presents conclusions.

## 2   Local Linear Models

Consider a nonlinear regression problem where a continuous response $y \in \mathbb{R}$ is to be predicted based on a $d$-dimensional predictor $x \in \mathbb{R}^d$. Let $D \equiv \{(x_t, y_t), t = 1, \ldots, T\}$ denote a set of training data. To estimate the conditional expectation $f(x_0) \equiv E[y|x_0]$, we consider the local linear expansion $f(x) \approx \alpha_0 + (x - x_0)'\beta_0$ in the neighborhood of $x_0$. In detail, we minimize the weighted least squares criterion

$$\mathcal{C}(\alpha, \beta; x_0) \equiv \sum_{t=1}^{T} (y_t - \alpha - (x_t - x_0)'\beta)^2 k(x_t, x_0) \qquad (1)$$

to determine estimates of the parameters $\alpha_0$ and $\beta_0$. Here $k(x_t, x_0)$ is a non-negative *weighting kernel* that assigns more weight to residuals in the neighborhood of $x_0$ than to residuals distant from $x_0$. In multivariate problems, a standard way of defining $k(x_t, x_0)$ is by applying a univariate, non-negative "mother kernel" $\phi(z)$ to the distance measure $\|x_t - x_0\|_\Omega \equiv \sqrt{(x_t - x_0)'\Omega(x_t - x_0)}$:

$$k(x_t, x_0) \equiv \frac{\phi(\|x_t - x_0\|_\Omega)}{\sum_{s=1}^{T} \phi(\|x_s - x_0\|_\Omega)}. \qquad (2)$$

Here $\Omega$ is a positive definite $d \times d$ matrix determining the relative importance assigned to different directions of the input space. For example, if $\phi(z)$ is a standard normal density, $k(x_t, x_0)$ is a normalized multivariate Gaussian with mean $x_0$ and covariance matrix $\Omega^{-1}$. Note that $k(x_t, x_0)$ is normalized so as to satisfy $\sum_{t=1}^{T} k(x_t, x_0) = 1$. Even though this restriction is not relevant directly with regard to the estimation of $\alpha_0$ and $\beta_0$, it will be needed in our discussion of entropic neighborhoods in Section 4.

Using the shorthand notation $\hat{\gamma}(x_0, \Omega) \equiv (\hat{\alpha}_0, \hat{\beta}_0')'$, the solution of the minimization problem (1) may be written conveniently as

$$\hat{\gamma}(x_0, \Omega) = (X'WX)^{-1}X'WY, \qquad (3)$$

where $X$ is the $T \times (d+1)$ design matrix with rows $(1, x_t' - x_0')'$, $Y$ is the vector of response values, and $W$ is a $T \times T$ diagonal matrix with entries $W_{t,t} = k(x_t, x_0)$. The resulting local linear fit at $x_0$ using the inverse covariance matrix $\Omega$ is simply $\hat{f}(x_0; \Omega) \equiv \hat{\alpha}_0$. Obviously, $\hat{f}(x_0; \Omega)$ depends on $\Omega$ through the definition of the weighting kernel (2). In the discussion below, our focus is on choices of $\Omega$ that lead to favorable estimates of the unknown function value $f(x_0)$.

## 3   Kernel Shaping

The local linear estimates resulting from different choices of $\Omega$ vary considerably in practice. A common strategy is to choose $\Omega$ proportional to the inverse sample covariance matrix. The remaining problem of finding the optimal scaling factor is equivalent to the problem of bandwidth selection in univariate smoothing [FG96, BBB99]. For example, the bandwidth is frequently chosen as a function of the distance between $x_0$ and its $k$th nearest neighbor in practical applications [FG96]. In this paper, we take a different viewpoint and argue that optimizing the "shape"

of the weighting kernel is at least as important as optimizing the bandwidth. More specifically, for a fixed "volume" of the weighting kernel, the bias of the estimate can be reduced drastically by shrinking the kernel in directions of large nonlinear variation of $f(x)$, and stretching it in directions of small nonlinear variation. This idea is illustrated using the example shown in Figure 1. The plotted function is sigmoidal along an index vector $\kappa$ and constant in directions orthogonal to $\kappa$. Therefore, a "shaped" weighting kernel is shrunk in the direction $\kappa$ and stretched orthogonally to $\kappa$, minimizing the exposure of the kernel to the nonlinear variation.

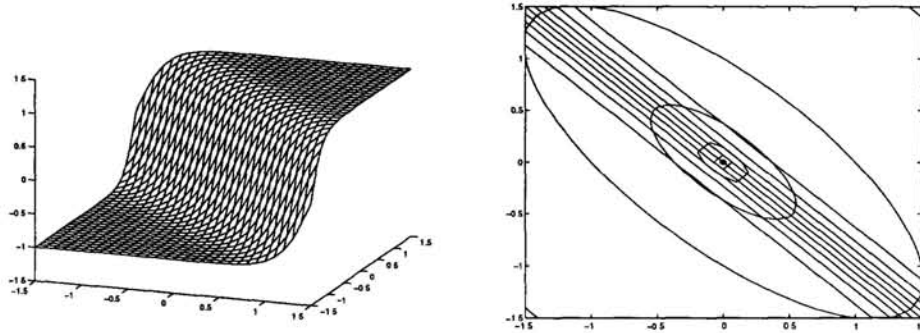

Figure 1:   *Left:* Example of a single index model of the form $y = g(x'\kappa)$ with $\kappa = (1,1)$ and $g(z) = \tanh(3z)$. *Right:* The contours of $g(z)$ are straight lines orthogonal to $\kappa$.

To distinguish formally the metric and the bandwidth of the weighting kernel, we rewrite $\Omega$ as follows:

$$\Omega \equiv \lambda \cdot (LL' + I). \tag{4}$$

Here $\lambda$ corresponds to the inverse bandwidth, and $L$ may be interpreted as a metric- or shape-matrix. Below we suggest an algorithm which is designed to minimize the bias with respect to the kernel metric. Clearly, for such an approach to be meaningful, we need to restrict the "volume" of the weighting kernel; otherwise, the bias of the estimate could be minimized trivially by choosing a zero bandwidth. For example, we might define $\lambda$ contingent on $L$ so as to satisfy $|\Omega| = c$ for some constant $c$. A serious disadvantage of this idea is that, by contrast to the nearest neighbor approach, $|\Omega|$ is independent of the design. As a more appropriate alternative, we define $\lambda$ in terms of a measure of the number of neighboring observations. In detail, we fix the volume of $k(x_t, x_0)$ in terms of the "entropy" of the weighting kernel. Then, we choose $\lambda$ so as to satisfy the resulting entropy constraint. Given this definition of the bandwidth, we determine the metric of $k(x_t, x_0)$ by minimizing the mean squared prediction error:

$$\mathcal{C}(L; D) \equiv \sum_{t=1}^{T} (y_t - f(x_t; \Omega))^2 \tag{5}$$

with respect to $L$. In this way, we obtain an approximation of the optimal kernel shape because the expectation of $\mathcal{C}(L; D)$ differs from the bias only by a variance term which is independent of $L$. Details of the entropic neighborhood criterion and of the numerical minimization procedure are described next.

## 4   Entropic Neighborhoods

We mentioned previously that, for a given shape matrix $L$, we choose the bandwidth parameter $\lambda$ in (4) so as to fulfill a volume constraint on the weighting kernel. For this purpose, we interpret the kernel weights $k(x_t, x_0)$ as probabilities. In particular,

as $k(x_t, x_0) > 0$ and $\sum_t k(x_t, x_0) = 1$ by definition (2), we can formulate the local *entropy* of $k(x_t, x_0)$:

$$H(\Omega) \equiv -\sum_{t=1}^{T} k(x_t, x_0) \log k(x_t, x_0). \tag{6}$$

The entropy of a probability distribution is typically thought of as a measure of uncertainty. In the context of the weighting kernel $k(x_t, x_0)$, $H(\Omega)$ can be used as a smooth measure of the "size" of the neighborhood that is used for averaging. To see this, note that in the extreme case where equal weights are placed on all observations in $D$, the entropy is maximized. At the other extreme, if the single nearest neighbor of $x_0$ is assigned the entire weight of one, the entropy attains its minimum value zero. Thus, fixing the entropy at a constant value $c$ is similar to fixing the number $k$ in the nearest neighbor approach. Besides justifying (6), the correspondence between $k$ and $c$ can also be used to derive a more intuitive volume parameter than the entropy level $c$. We specify $c$ in terms of a hypothetical weighting kernel that places equal weight on the $k$ nearest neighbors of $x_0$ and zero weight on the remaining observations. Note that the entropy of this hypothetical kernel is $\log k$. Thus, it is natural to characterize the size of an entropic neighborhood in terms of $k$, and then to determine $\lambda$ by numerically solving the nonlinear equation system (for details, see [OH99])

$$H(\Omega) = \log k. \tag{7}$$

More precisely, we report the number of neighbors in terms of the *equivalent sample fraction* $\rho \equiv k/T$ to further intuition. This idea is illustrated in Figure 2 using a one- and a two-dimensional example. The equivalent sample fractions are $\rho = 30\%$ and $\rho = 50\%$, respectively. Note that in both cases the weighting kernel is wider in regions with few observations, and narrower in regions with many observations. As a consequence, the number of observations within contours of equal weighting remains approximately constant across the input space.

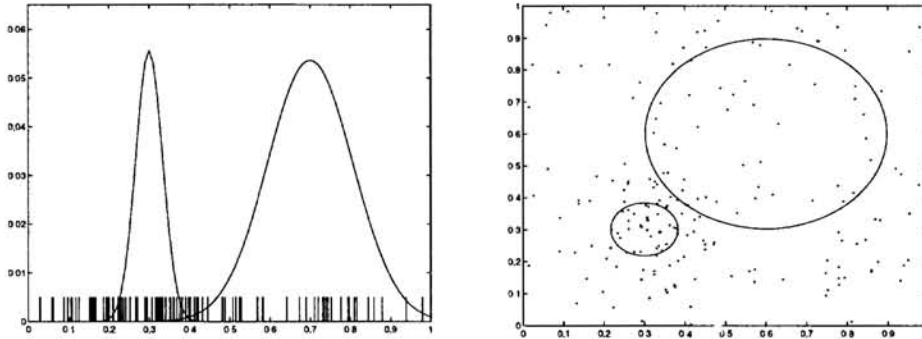

Figure 2: *Left:* Univariate weighting kernel $k(\cdot, x_0)$ evaluated at $x_0 = 0.3$ and $x_0 = 0.7$ based on a sample data set of 100 observations (indicated by the bars at the bottom). *Right:* Multivariate weighting kernel $k(\cdot, x_0)$ based on a sample data set of 200 observations. The two ellipsoids correspond to 95% contours of a weighting kernel evaluated at $(0.3, 0.3)'$ and $(0.6, 0.6)'$.

To summarize, we define the value of $\lambda$ by fixing the equivalent sample fraction parameter $\rho$, and subsequently minimize the prediction error on the training set with respect to the shape matrix $L$. Note that we allow for the possibility that $L$ may be of reduced rank $l \leq d$ as a means of controlling the number of free parameters. As a minimization procedure, we use a variant of gradient descent that

accounts for the entropy constraint. In particular, our algorithm relies on the fact that (7) is differentiable with respect to $L$. Due to space limitations, the interested reader is referred to [OH99] for a formal derivation of the involved gradients and for a detailed description of the optimization procedure.

## 5   Experiments

In this section we compare kernel shaping to standard local linear regression using a fixed spherical kernel in two examples. First, we evaluate the performance using a simple toy problem which allows us to estimate confidence intervals for the prediction accuracy using Monte Carlo simulation. Second, we investigate a data set from the machine learning data base at UC Irvine [BKM98].

### 5.1   Mexican Hat Function

In our first example, we employ Monte Carlo simulation to evaluate the performance of kernel shaping in a five-dimensional regression problem. For this purpose, 20 sets of 500 data points each are generated independently according to the model

$$y = \cos(5\sqrt{x_1^2 + x_2^2}) \cdot \exp(-(x_1^2 + x_2^2)). \tag{8}$$

Here the predictor variables $x_1, \ldots, x_5$ are drawn according to a five-dimensional standard normal distribution. Note that, even though the regression is carried out in a five-dimensional predictor space, $y$ is really only a function of the variables $x_1$ and $x_2$. In particular, as dimensions two through five do not contribute any information with regard to the value of $y$, kernel shaping should effectively discard these variables. Note also that there is no noise in this example.

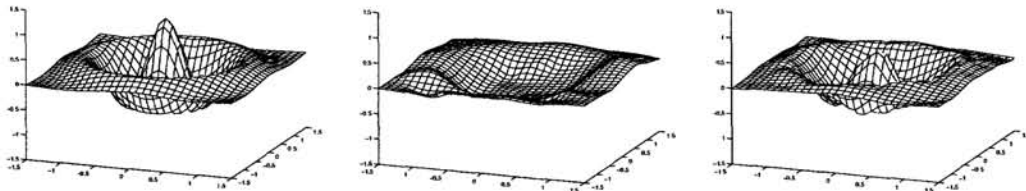

Figure 3:   *Left:* "True" Mexican hat function. *Middle:* Local linear estimate using a spherical kernel ($\rho = 2\%$). *Right:* Local linear estimate using kernel shaping ($\rho = 2\%$). Both estimates are based on a training set consisting of 500 data points.

Figure 3 shows a plot of the true function, the spherical estimate, and the estimate using kernel shaping as functions of $x_1$ and $x_2$. The true function has the familiar "Mexican hat" shape, which is recovered by the estimates to different degrees. We evaluate the local linear estimates for values of the equivalent neighborhood fraction parameter $\rho$ in the range from 1% to 15%. Note that, to warrant a fair comparison, we used the entropic neighborhood also to determine the bandwith of the spherical estimate. For each value of $\rho$, 20 models are estimated using the 20 artificially generated training sets, and subsequently their performance is evaluated on the training set and on the test set of $31 \times 31$ grid points shown in Figure 3. The shape matrix $L$ has maximal rank $l = 5$ in this experiment. Our results for local linear regression using the spherical kernel and kernel shaping are summarized in Table 1. Performance is measured in terms of the mean $R^2$-value of the 20 models, and standard deviations are reported in parenthesis.

| Algorithm | | Training $R^2$ | Test $R^2$ |
|---|---|---|---|
| spherical kernel | $\rho = 1\%$ | 0.961 (0.005) | 0.215 (0.126) |
| spherical kernel | $\rho = 2\%$ | 0.871 (0.014) | 0.293 (0.082) |
| spherical kernel | $\rho = 5\%$ | 0.680 (0.029) | 0.265 (0.043) |
| spherical kernel | $\rho = 10\%$ | 0.507 (0.038) | 0.213 (0.030) |
| spherical kernel | $\rho = 20\%$ | 0.341 (0.039) | 0.164 (0.021) |
| kernel shaping | $\rho = 1\%$ | 0.995 (0.001) | 0.882 (0.024) |
| kernel shaping | $\rho = 2\%$ | 0.984 (0.002) | 0.909 (0.017) |
| kernel shaping | $\rho = 5\%$ | 0.923 (0.009) | 0.836 (0.023) |
| kernel shaping | $\rho = 15\%$ | 0.628 (0.035) | 0.517 (0.035) |

Table 1: Performances in the toy problem. The results for kernel shaping were obtained using 200 gradient descent steps with step size $\alpha = 0.2$.

The results in Table 1 indicate that the optimal performance on the test set is obtained using the parameter values $\rho = 2\%$ both for kernel shaping ($R^2 = 0.909$) and for the spherical kernel ($R^2 = 0.293$). Given the large difference between the $R^2$ values, we conclude that kernel shaping clearly outperforms the spherical kernel on this data set.

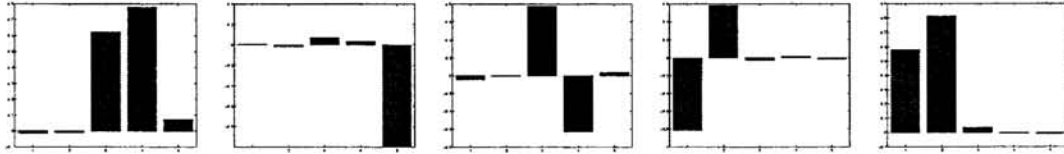

Figure 4: The eigenvectors of the estimate of $\Omega$ obtained on the first of 20 training sets. The graphs are ordered from left to right by increasing eigenvalues (decreasing extension of the kernel in that direction): 0.76, 0.76, 0.76, 33.24, 34.88.

Finally, Figure 4 shows the eigenvectors of the optimized $\Omega$ on the first of the 20 training sets. The eigenvectors are arranged according to the size of the corresponding eigenvalues. Note that the two rightmost eigenvectors, which correspond to the directions of minimum kernel extension, span exactly the $x_1$-$x_2$-space where the true function lives. The kernel is stretched in the remaining directions, effectively discarding nonlinear contributions from $x_3$, $x_4$, and $x_5$.

## 5.2 Abalone Database

The task in our second example is to predict the age of abalone based on several measurements. More specifically, the response variable is obtained by counting the number of rings in the shell in a time-consuming procedure. Preferably, the age of the abalone could be predicted from alternative measurements that may be obtained more easily. In the data set, eight candidate measurements including sex, dimensions, and various weights are reported along with the number of rings of the abalone as predictor variables. We normalize these variables to zero mean and unit variance prior to estimation. Overall, the data set consists of 4177 observations. To prevent possible artifacts resulting from the order of the data records, we randomly draw 2784 observations as a training set and use the remaining 1393 observations as a test set. Our results are summarized in Table 2 using various settings for the rank $l$, the equivalent fraction parameter $\rho$, and the gradient descent step size $\alpha$. The optimal choice for $\rho$ is 20% both for kernel shaping ($R^2 = 0.582$) and for the spherical kernel ($R^2 = 0.572$). Note that the performance improvement due to kernel shaping is negligible in this experiment.

| Kernel | | Training $R^2$ | Test $R^2$ |
|---|---|---|---|
| spherical kernel | $\rho = 0.05$ | 0.752 | 0.543 |
| spherical kernel | $\rho = 0.10$ | 0.686 | 0.564 |
| spherical kernel | $\rho = 0.20$ | 0.639 | 0.572 |
| spherical kernel | $\rho = 0.50$ | 0.595 | 0.565 |
| spherical kernel | $\rho = 0.70$ | 0.581 | 0.552 |
| spherical kernel | $\rho = 0.90$ | 0.568 | 0.533 |
| kernel shaping | $l = 5, \rho = 0.20, \alpha = 0.5$ | 0.705 | 0.575 |
| kernel shaping | $l = 5, \rho = 0.20, \alpha = 0.2$ | 0.698 | 0.577 |
| kernel shaping | $l = 2, \rho = 0.10, \alpha = 0.2$ | 0.729 | 0.574 |
| kernel shaping | $l = 2, \rho = 0.20, \alpha = 0.2$ | 0.663 | 0.582 |
| kernel shaping | $l = 2, \rho = 0.50, \alpha = 0.2$ | 0.603 | 0.571 |
| kernel shaping | $l = 2, \rho = 0.20, \alpha = 0.5$ | 0.669 | 0.582 |

Table 2: Results using the Abalone database after 200 gradient descent steps.

## 6    Conclusions

We introduced a data-driven method to improve the performance of local linear estimates in high dimensions by optimizing the shape of the weighting kernel. In our experiments we found that kernel shaping clearly outperformed local linear regression using a spherical kernel in a five-dimensional toy example, and led to a small performance improvement in a second, real-world example. To explain the results of the second experiment, we note that kernel shaping aims at exploiting global structure in the data. Thus, the absence of a larger performance improvement may suggest simply that no corresponding structure prevails in that data set. That is, even though optimal kernel shapes exist *locally*, they may vary accross the predictor space so that they cannot be approximated by any particular *global* shape. Preliminary experiments using a localized variant of kernel shaping did not lead to significant performance improvements in our experiments.

## Acknowledgments

The work of Dirk Ormoneit was supported by a grant of the Deutsche Forschungsgemeinschaft (DFG) as part of its post-doctoral program. Trevor Hastie was partially supported by NSF grant DMS-9803645 and NIH grant ROI-CA-72028-01. Carrie Grimes pointed us to misleading formulations in earlier drafts of this work.

## References

[AMS97]  C. G. Atkeson, A. W. Moore, and S. Schaal. Locally weighted learning. *Artificial Intelligence Review*, 11:11–73, 1997.

[BBB99]  M. Birattari, G. Bontempi, and H. Bersini. Lazy learning meets the recursive least squares algorithm. In M. J. Kearns, S. A. Solla, and D. A. Cohn, editors, *Advances in Neural Information Processing Systems 11*. The MIT Press, 1999.

[BKM98]  C. Blake, E. Koegh, and C. J. Merz. UCI Repository of machine learning databases. http://www.ics.uci.edu/~mlearn/MLRepository.html.

[Cle79]  W. S. Cleveland. Robust locally weighted regression and smoothing scatterplots. *Journal of the American Statistical Association*, 74:829–836, 1979.

[FG96]  J. Fan and I. Gijbels. *Local Polynomial Modelling and Its Applications*. Chapman & Hall, 1996.

[OH99]  D. Ormoneit and T. Hastie. Optimal kernel shapes for local linear regression. Tech. report 1999-11, Department of Statistics, Stanford University, 1999.